# Receding Horizon
# Differential Dynamic Programming

**Yuval Tassa** *                    **Tom Erez & Bill Smart** †

## Abstract

The control of high-dimensional, continuous, non-linear dynamical systems is a key problem in reinforcement learning and control. Local, trajectory-based methods, using techniques such as Differential Dynamic Programming (DDP), are not directly subject to the curse of dimensionality, but generate only local controllers. In this paper,we introduce Receding Horizon DDP (RH-DDP), an extension to the classic DDP algorithm, which allows us to construct stable and robust controllers based on a library of local-control trajectories. We demonstrate the effectiveness of our approach on a series of high-dimensional problems using a simulated multi-link swimming robot. These experiments show that our approach effectively circumvents dimensionality issues, and is capable of dealing with problems of (at least) 24 state and 9 action dimensions.

## 1   Introduction

We are interested in learning controllers for high-dimensional, highly non-linear dynamical systems, continuous in state, action, and time. Local, trajectory-based methods, using techniques such as Differential Dynamic Programming (DDP), are an active field of research in the Reinforcement Learning and Control communities. Local methods do not model the value function or policy over the entire state space by focusing computational effort along likely trajectories. Featuring algorithmic complexity polynomial in the dimension, local methods are not directly affected by dimensionality issues as space-filling methods.

In this paper, we introduce Receding Horizon DDP (RH-DDP), a set of modifications to the classic DDP algorithm, which allows us to construct stable and robust controllers based on local-control trajectories in highly non-linear, high-dimensional domains. Our new algorithm is reminiscent of Model Predictive Control, and enables us to form a time-independent value function approximation along a trajectory. We aggregate several such trajectories into a library of locally-optimal linear controllers which we then select from, using a nearest-neighbor rule.

Although we present several algorithmic contributions, a main aspect of this paper is a conceptual one. Unlike much of recent related work (below), we are not interested in learning to follow a pre-supplied reference trajectory. We define a reward function which represents a global measure of performance relative to a high level objective, such as swimming towards a target. Rather than a reward based on distance from a given desired configuration, a notion which has its roots in the control community's definition of the problem, this global reward dispenses with a "path planning" component and requires the controller to solve the entire problem.

We demonstrate the utility of our approach by learning controllers for a high-dimensional simulation of a planar, multi-link swimming robot. The *swimmer* is a model of an actuated chain of links in a viscous medium, with two location and velocity coordinate pairs, and an angle and angular velocity for each link. The controller must determine the applied torque, one action dimension for

each articulated joint. We reward controllers that cause the swimmer to swim to a target, brake on approach and come to a stop over it.

We synthesize controllers for several swimmers, with state dimensions ranging from 10 to 24 dimensions. The controllers are shown to exhibit complex locomotive behaviour in response to real-time simulated interaction with a user-controlled target.

## 1.1 Related work

Optimal control of continuous non-linear dynamical systems is a central research goal of the RL community. Even when important ingredients such as stochasticity and on-line learning are removed, the exponential dependence of computational complexity on the dimensionality of the domain remains a major computational obstacle. Methods designed to alleviate the curse of dimensionality include adaptive discretizations of the state space [1], and various domain-specific manipulations [2] which reduce the effective dimensionality.

Local trajectory-based methods such as DDP were introduced to the NIPS community in [3], where a local-global hybrid method is employed. Although DDP is used there, it is considered an aid to the global approximator, and the local controllers are constant rather than locally-linear. In this decade DDP was reintroduced by several authors. In [4] the idea of using the second order local DDP models to make locally-linear controllers is introduced. In [5] DDP was applied to the challenging high-dimensional domain of autonomous helicopter control, using a reference trajectory. In [6] a minimax variant of DDP is used to learn a controller for bipedal walking, again by designing a reference trajectory and rewarding the walker for tracking it. In [7], trajectory-based methods including DDP are examined as possible models for biological nervous systems. Local methods have also been used for purely policy-based algorithms [8, 9, 10], without explicit representation of the value function.

The best known work regarding the swimming domain is that by Ijspeert and colleagues (e.g. [11]) using Central Pattern Generators. While the inherently stable domain of swimming allows for such open-loop control schemes, articulated complex behaviours such as turning and tracking necessitate full feedback control which CPGs do not provide.

## 2 Methods

### 2.1 Definition of the problem

We consider the discrete-time dynamics $x^{k+1} = F(x^k, u^k)$ with states $x \in \mathbb{R}^n$ and actions $u \in \mathbb{R}^m$. In this context we assume $F(x^k, u^k) = x^k + \int_0^{\Delta t} f(x(t), u^k) dt$ for a continuous $f$ and a small $\Delta t$, approximating the continuous problem and identifying with it in the $\Delta t \to 0$ limit. Given some scalar reward function $r(x, u)$ and a fixed initial state $x^1$ (superscripts indicating the time index), we wish to find the policy which maximizes the total reward[1] acquired over a finite temporal horizon:

$$\pi^*(x^k, k) = \underset{\pi(\cdot, \cdot)}{\operatorname{argmax}} [\sum_{i=k}^{N} r(x^i, \pi(x^i, i))].$$

The quantity maximized on the RHS is the *value function*, which solves Bellman's equation:

$$V(x, k) = \max_u [r(x, u) + V(F(x, u), k+1)]. \tag{1}$$

Each of the functions in the sequence $\{V(x, k)\}_{k=1}^{N}$ describes the optimal reward-to-go of the *optimization subproblem* from $k$ to $N$. This is a manifestation of the *dynamic programming principle*. If $N = \infty$, essentially eliminating the distinction between different time-steps, the sequence collapses to a global, time-independent value function $V(x)$.

### 2.2 DDP

Differential Dynamic Programming [12, 13] is an iterative improvement scheme which finds a locally-optimal trajectory emanating from a fixed starting point $x^1$. At every iteration, an approx-

imation to the time-dependent value function is constructed along the current trajectory $\{x^k\}_{k=1}^N$, which is formed by iterative application of $F$ using the current control sequence $\{u^k\}_{k=1}^N$. Every iteration is comprised of two sweeps of the trajectory: a *backward* and a *forward* sweep.

In the *backward sweep*, we proceed backwards in time to generate local models of $V$ in the following manner. Given quadratic models of $V(x^{k+1}, k+1)$, $F(x^k, u^k)$ and $r(x^k, u^k)$, we can approximate the *unmaximised* value function, or $Q$-function,

$$Q(x^k, u^k) = r(x^k, u^k) + V^{k+1}(F(x^k, u^k)) \tag{2}$$

as a quadratic model around the present state-action pair $(x^k, u^k)$:

$$Q(x^k + \delta x, u^k + \delta u) \approx Q_0 + Q_x \delta x + Q_u \delta u + \frac{1}{2}[\delta x^T \ \delta u^T] \begin{bmatrix} Q_{xx} & Q_{xu} \\ Q_{ux} & Q_{uu} \end{bmatrix} \begin{bmatrix} \delta x \\ \delta u \end{bmatrix} \tag{3}$$

Where the coefficients $Q_{\star\star}$ are computed by equating coefficients of similar powers in the second-order expansion of (2)

$$
\begin{aligned}
Q_x &= r_x + V_x^{k+1} F_x^k & Q_{xx} &= r_{xx} + F_x^k V_{xx}^{k+1} F_x^k + V_x^{k+1} F_{xx}^k \\
Q_u &= r_u + V_x^{k+1} F_u^k & Q_{uu} &= r_{uu} + F_u^k V_{xx}^{k+1} F_u^k + V_x^{k+1} F_{uu}^k \\
& & Q_{xu} &= r_{xu} + F_x^k V_{xx}^{k+1} F_u^k + V_x^{k+1} F_{xu}^k.
\end{aligned}
\tag{4}
$$

Once the local model of $Q$ is obtained, the maximizing $\delta u$ is solved for

$$\delta u^* = \underset{\delta u}{\operatorname{argmax}}[Q(x^k + \delta x, u^k + \delta u)] = -Q_{uu}^{-1}(Q_u + Q_{ux}\delta x) \tag{5}$$

and plugged back into (3) to obtain a quadratic approximation of $V^k$:

$$V_0^k = V_0^{k+1} - Q_u(Q_{uu})^{-1} Q_u \tag{6a}$$

$$V_x^k = Q_x^{k+1} - Q_u(Q_{uu})^{-1} Q_{ux} \tag{6b}$$

$$V_{xx}^k = Q_{xx}^{k+1} - Q_{xu}(Q_{uu})^{-1} Q_{ux}. \tag{6c}$$

This quadratic model can now serve to propagate the approximation to $V^{k-1}$. Thus, equations (4), (5) and (6) iterate in the backward sweep, computing a local model of the Value function along with a modification to the policy in the form of an open-loop term $-Q_{uu}^{-1}Q_u$ and a feedback term $-Q_{uu}^{-1}Q_{ux}\delta x$, essentially solving a local linear-quadratic problem in each step. In some senses, DDP can be viewed as dual to the Extended Kalman Filter (though employing a higher order expansion of $F$).

In the *forward sweep* of the DDP iteration, both the open-loop and feedback terms are combined to create a new control sequence $(\hat{u}^k)_{k=1}^N$ which results in a new nominal trajectory $(\hat{x}^k)_{k=1}^N$.

$$\hat{x}^1 = x^1 \tag{7a}$$

$$\hat{u}^k = u^k - Q_{uu}^{-1}Q_u - Q_{uu}^{-1}Q_{ux}(\hat{x}^k - x^k) \tag{7b}$$

$$\hat{x}^{k+1} = F(\hat{x}^k, \hat{u}^k) \tag{7c}$$

We note that in practice the inversion in (5) must be conditioned. We use a Levenberg Marquardt-like scheme similar to the ones proposed in [14]. Similarly, the $u$-update in (7b) is performed with an adaptive line search scheme similar to the ones described in [15].

### 2.2.1 Complexity and convergence

The leading complexity term of one iteration of DDP itself, assuming the model of $F$ as required for (4) is given, is $O(Nm^{\gamma_1})$ for computing (6) $N$ times, with $2 < \gamma_1 < 3$, the complexity-exponent of inverting $Q_{uu}$. In practice, the greater part of the computational effort is devoted to the measurement of the dynamical quantities in (4) or in the propagation of collocation vectors as described below.

DDP is a second order algorithm with convergence properties similar to, or better than Newton's method performed on the full vectorial $u^k$ with an exact $Nm \times Nm$ Hessian [16]. In practice, convergence can be expected after 10-100 iterations, with the stopping criterion easily determined as the size of the policy update plummets near the minimum.

### 2.2.2 Collocation Vectors

We use a new method of obtaining the quadratic model of $Q$ (Eq. (2)), inspired by [17][2]. Instead of using (4), we fit this quadratic model to samples of the value function at a cloud of collocation vectors $\{x_i^k, u_i^k\}_{i=1..p}$, spanning the neighborhood of every state-action pair along the trajectory. We can directly measure $r(x_i^k, u_i^k)$ and $F(x_i^k, u_i^k)$ for each point in the cloud, and by using the approximated value function at the next time step, we can estimate the value of (2) at every point:

$$q(x_i^k, u_i^k) = r(x_i^k, u_i^k) + V^{k+1}(F(x_i^k, u_i^k))$$

Then, we can insert the values of $q(x_i^k, u_i^k)$ and $(x_i^k, u_i^k)$ on the LHS and RHS of (3) respectively, and solve this set of $p$ linear equations for the $Q_{\star\star}$ terms. If $p > (3(n + m) + (m + n)^2)/2$, and the cloud is in general configuration, the equations are non-singular and can be easily solved by a generic linear algebra package.

There are several advantages to using such a scheme. The full nonlinear model of $F$ is used to construct $Q$, rather than only a second-order approximation. $F_{xx}$, which is an $n \times n \times n$ tensor need not be stored. The addition of more vectors can allow the modeling of noise, as suggested in [17]. In addition, this method allows us to more easily apply general coordinate transformations in order to represent $V$ in some internal space, perhaps of lower dimension.

The main drawback of this scheme is the additional complexity of an $O(Np^{\gamma_2})$ term for solving the $p$-equation linear system. Because we can choose $\{x_i^k, u_i^k\}$ in way which makes the linear system sparse, we can enjoy the $\gamma_2 < \gamma_1$ of sparse methods and, at least for the experiments performed here, increase the running time only by a small factor.

In the same manner that DDP is dually reminiscent of the Extended Kalman Filter, this method bears a resemblance to the test vectors propagated in the Unscented Kalman Filter [18], although we use a quadratic, rather than linear number of collocation vectors.

### 2.3 Receding Horizon DDP

When seeking to synthesize a global controller from many local controllers, it is essential that the different local components operate synergistically. In our context this means that local models of the value function must all model the same function, which is not the case for the standard DDP solution. The local quadratic models which DDP computes around the trajectory are approximations to $V(x, k)$, the time-dependent value function. The standard method in RL for creating a global value function is to use an exponentially discounted horizon. Here we propose a fixed-length non-discounted Receding Horizon scheme in the spirit of Model Predictive Control [19].

Having computed a DDP solution to some problem starting from many different starting points $x_1$, we can discard all the models computed for points $x^{k>1}$ and save only the ones around the $x_1$'s. Although in this way we could accumulate a time-independent approximation to $V(x, N)$ only, starting each run of $N$-step DDP from scratch would be prohibitively expensive. We therefore propose the following: After obtaining the solution starting from $x^1$, we save the local model at $k = 1$ and proceed to solve a new $N$-step problem starting at $x^2$, this time initialized with the policy obtained on the previous run, shifted by one time-step, and appended with the last control $u_{new} = [u^2, u^3...u^N u^N]$. Because this control sequence is very close to the optimal solution, the second-order convergence of DDP is in full effect and the algorithm converges in 1 or 2 sweeps. Again saving the model at the first time step, we iterate. We stress the that without the fast and exact convergence properties of DDP near the maximum, this algorithm would be far less effective.

### 2.4 Nearest Neighbor control with Trajectory Library

A run of DDP computes a locally quadratic model of $V$ and a locally linear model of $u$, expressed by the gain term $-Q_{uu}^{-1}Q_{ux}$. This term generalizes the open-loop policy to a tube around the trajectory, inside of which a basin-of-attraction is formed. Having lost the dependency on the time $k$ with the receding-horizon scheme, we need some space-based method of determining which local gain model we select at a given state. The simplest choice, which we use here, is to select the nearest Euclidian neighbor.

Outside of the basin-of-attraction of a single trajectory, we can expect the policy to perform very poorly and lead to numerical divergence if no constraint on the size of $u$ is enforced. A possible solution to this problem is to fill some volume of the state space with a library of local-control trajectories [20], and consider all of them when selecting the nearest linear gain model.

# 3 Experiments

## 3.1 The *swimmer* dynamical system

We describe a variation of the *d-link swimmer* dynamical system [21]. A stick or *link* of length $l$, lying in a plane at an angle $\theta$ to some direction, parallel to $\hat{\mathbf{t}} = \left( \begin{smallmatrix} \cos(\theta) \\ \sin(\theta) \end{smallmatrix} \right)$ and perpendicular to $\hat{\mathbf{n}} = \left( \begin{smallmatrix} -\sin(\theta) \\ \cos(\theta) \end{smallmatrix} \right)$, moving with velocity $\dot{\mathbf{x}}$ in a viscous fluid, is postulated to admit a normal frictional force $-k_n l \hat{\mathbf{n}} (\dot{\mathbf{x}} \cdot \hat{\mathbf{n}})$ and a tangential frictional force $-k_t l \hat{\mathbf{t}} (\dot{\mathbf{x}} \cdot \hat{\mathbf{t}})$, with $k_n > k_t > 0$. The swimmer is modeled as a chain of $d$ such links of lengths $l_i$ and masses $m_i$, its configuration described by the generalized coordinates $\mathbf{q} = \left( \begin{smallmatrix} \mathbf{x}_{cm} \\ \boldsymbol{\theta} \end{smallmatrix} \right)$, of two center-of-mass coordinates and $d$ angles. Letting $\bar{\mathbf{x}}_i = \mathbf{x}_i - \mathbf{x}_{cm}$ be the positions of the link centers WRT the center of mass , the Lagrangian is

$$L = \tfrac{1}{2} \dot{\mathbf{x}}_{cm}^2 \sum_i m_i + \tfrac{1}{2} \sum_i m_i \dot{\bar{\mathbf{x}}}_i^2 + \tfrac{1}{2} \sum_i I_i \dot{\theta}_i^2$$

with $I_i = \frac{1}{12} m_i l_i^2$ the moments-of-inertia. The relationship between the relative position vectors and angles of the links is given by the $d-1$ equations $\bar{\mathbf{x}}_{i+1} - \bar{\mathbf{x}}_i = \frac{1}{2} l_{i+1} \hat{\mathbf{t}}_{i+1} + \frac{1}{2} l_i \hat{\mathbf{t}}_i$, which express the joining of successive links, and by the equation $\sum_i m_i \bar{\mathbf{x}}_i = 0$ which comes from the

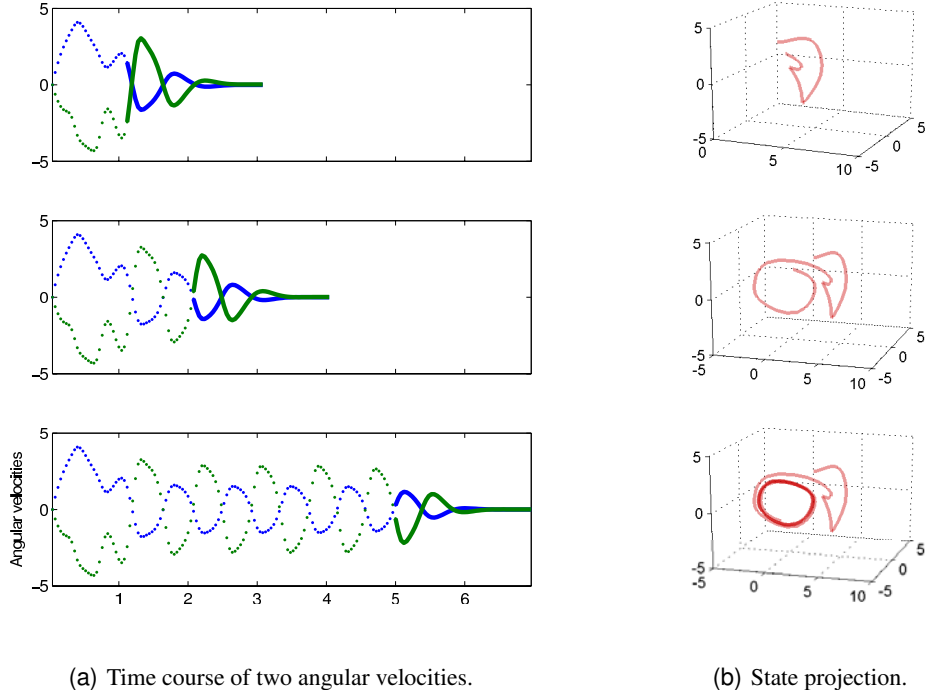

(a) Time course of two angular velocities.      (b) State projection.

Figure 1: RH-DDP trajectories. (a) three snapshots of the receding horizon trajectory (dotted) with the current finite-horizon optimal trajectory (solid) appended, for two state dimensions. (b) Projections of the same receding-horizon trajectories onto the largest three eigenvectors of the full state covariance matrix. As described in Section 3.3, the linear regime of the reward, here applied to a 3-swimmer, compels the RH trajectories to a steady swimming gait – a limit cycle.

definition of the $\bar{x}_i$'s relative to the center-of-mass. The function

$$F = -\tfrac{1}{2}k_n \sum_i [l_i(\dot{\mathbf{x}}_i \cdot \hat{\mathbf{n}}_i)^2 + \tfrac{1}{12}l_i^3\dot{\theta}_i^2] - \tfrac{1}{2}k_t \sum_i l_i(\dot{\mathbf{x}}_i \cdot \hat{\mathbf{t}}_i)^2$$

known as the *dissipation function*, is that function whose derivatives WRT the $\dot{q}_i$'s provide the postulated frictional forces. With these in place, we can obtain $\ddot{\mathbf{q}}$ from the $2+d$ Euler-Lagrange equations:

$$\tfrac{d}{dt}(\tfrac{\partial}{\partial q_i}L) = \tfrac{\partial}{\partial \dot{q}_i}F + \mathbf{u}$$

with $\mathbf{u}$ being the external forces and torques applied to the system. By applying $d-1$ torques $\tau_j$ in action-reaction pairs at the joints $u_i = \tau_i - \tau_{i-1}$, the isolated nature of the dynamical system is preserved. Performing the differentiations, solving for $\ddot{\mathbf{q}}$, and letting $\mathbf{x} = \left(\begin{smallmatrix}\mathbf{q}\\\dot{\mathbf{q}}\end{smallmatrix}\right)$ be the $4+2d$-dimensional state variable, finally gives the dynamics $\dot{\mathbf{x}} = \left(\begin{smallmatrix}\dot{\mathbf{q}}\\\ddot{\mathbf{q}}\end{smallmatrix}\right) = f(\mathbf{x}, \mathbf{u})$.

## 3.2  Internal coordinates

The two coordinates specifying the position of the center-of-mass and the $d$ angles are defined relative to an external coordinate system, which the controller should not have access to. We make a coordinate transformation into *internal* coordinates, where only the $d-1$ relative angles $\{\hat{\theta}_j = \theta_{j+1} - \theta_j\}_{j=1}^{d-1}$ are given, and the location of the target is given relative to coordinate system fixed on one of the links. This makes the learning isotropic and independent of a specific location on the plane. The collocation method allows us to perform this transformation directly on the vector cloud without having to explicitly differentiate it, as we would have had to using classical DDP. Note also that this transformation reduces the dimension of the state (one angle less), suggesting the possibility of further dimensionality reduction.

## 3.3  The reward function

The reward function we used was

$$r(x, u) = -c_x \frac{||x_{nose}||^2}{\sqrt{||x_{nose}||^2 + 1}} - c_u||u||^2 \tag{8}$$

Where $x_{nose} = [x_1 x_2]^\mathsf{T}$ is the 2-vector from some designated point on the swimmer's body to the target (the origin in internal space), and $c_x$ and $c_u$ are positive constants. This reward is maximized when the nose is brought to rest on the target under a quadratic action-cost penalty. It should not be confused with the *desired state* reward of classical optimal control since values are specified only for 2 out of the $2d+4$ coordinates. The functional form of the target-reward term is designed to be linear in $||x_{nose}||$ when far from the target and quadratic when close to it (Figure 2(b)). Because

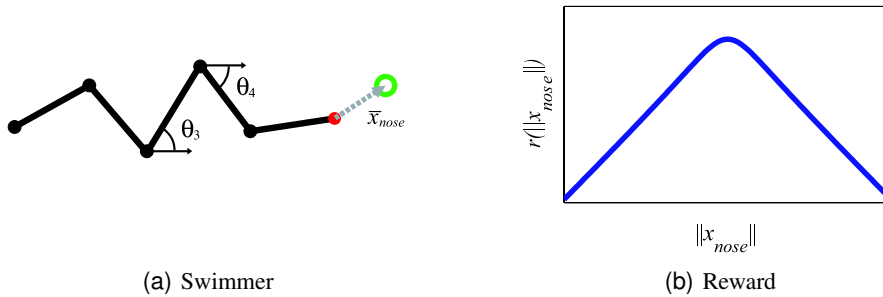

(a) Swimmer                                        (b) Reward

Figure 2: (a) A *5-swimmer* with the "nose" point at its tip and a ring-shaped target. (b) The functional form of the planar reward component $r(x_{nose}) = -||x_{nose}||^2/\sqrt{||x_{nose}||^2 + 1}$. This form translates into a steady swimming gait at large distances with a smooth braking and stopping at the goal.

of the differentiation in Eq. (5), the solution is independent of $V_0$, the constant part of the value. Therefore, in the linear regime of the reward function, the solution is independent of the distance from the target, and all the trajectories are quickly compelled to converge to a one-dimensional manifold in state-space which describes steady-state swimming (Figure 1(b)). Upon nearing the target, the swimmer must initiate a braking maneuver, and bring the nose to a standstill over the target. For targets that are near the swimmer, the behaviour must also include various turns and jerks, quite different from steady-state swimming, which maneuver the nose into contact with the target. Our experience during interaction with the controller, as detailed below, leads us to believe that the behavioral variety that would be exhibited by a hypothetical exact optimal controller for this system to be extremely large.

## 4  Results

In order to asses the controllers we constructed a real-time interaction package[3]. By dragging the target with a cursor, a user can interact with controlled swimmers of 3 to 10 links with a state dimension varying from 10 to 24, respectively. Even with controllers composed of a single trajectory, the swimmers perform quite well, turning, tracking and braking on approach to the target.

All of the controllers in the package control swimmers with unit link lengths and unit masses. The normal-to-tangential drag coefficient ratio was $k_n/k_t = 25$. The function $F$ computes a single 4th-order Runge-Kutta integration step of the continuous dynamics $F(x^k, u^k) = x^k + \int_t^{t+\Delta t} f(x^k, u^k) dt$ with $\Delta t = 0.05_s$. The receding horizon window was of 40 time-steps, or 2 seconds.

When the state doesn't gravitate to one of the basins of attraction around the trajectories, numerical divergence can occur. This effect can be initiated by the user by quickly moving the target to a "surprising" location. Because nonlinear viscosity effects are not modeled and the local controllers are also linear, exponentially diverging torques and angular velocities can be produced. When adding as few as 20 additional trajectories, divergence is almost completely avoided.

Another claim which may be made is that there is no guarantee that the solutions obtained, even on the trajectories, are in fact optimal. Because DDP is a local optimization method, it is bound to stop in a local minimum. An extension of this claim is that even if the solutions are optimal, this has to do with the swimmer domain itself, which might be inherently convex in some sense and therefore an "easy" problem.

While both divergence and local minima are serious issues, they can both be addressed by appealing to our panoramic motivation in the biology. Real organisms cannot apply unbounded torque. By hard-limiting the torque to large but finite values, non-divergence can be guaranteed[4]. Similarly, local minima exist even in the motor behaviour of the most complex organisms, famously evidenced by Fosbury's reinvention of the high jump.

Regarding the easiness or difficulty of the swimmer problem – we made the documented code available and hope that it might serve as a useful benchmark for other algorithms.

## 5  Conclusions

The significance of this work lies at its outlining of a new kind of tradeoff in nonlinear motor control design. If biological realism is an accepted design goal, and physical and biological constraints taken into account, then the expectations we have from our controllers can be more relaxed than those of the control engineer. The unavoidable eventual failure of any specific biological organism makes the design of truly robust controllers a futile endeavor, in effect putting more weight on the mode, rather than the tail of the behavioral distribution. In return for this forfeiture of global guarantees, we gain very high performance in a small but very dense sub-manifold of the state-space.

Since we make use of biologically grounded arguments, we briefly outline the possible implications of this work to biological nervous systems. It is commonly acknowledged, due both to theoretical arguments and empirical findings, that some form of dimensionality reduction must be at work in neural control mechanisms. A common object in models which attempt to describe this reduction is the *motor primitive*, a hypothesized *atomic* motor program which is combined with other such programs in a small "alphabet", to produce complex behaviors in a given context. Our controllers imply a different reduction: a set of complex prototypical motor programs, each of which is near-optimal only in a small volume of the state-space, yet in that space describes the entire complexity of the solution. Giving the simplest building blocks of the model such a high degree of task specificity or context, would imply a very large number of these *motor prototypes* in a real nervous system, an order of magnitude analogous, in our linguistic metaphor, to that of words and concepts.

## Footnotes

*Y. Tassa is with the Hebrew University, Jerusalem, Israel.

†T. Erez and W.D. Smart are with the Washington University in St. Louis, MO, USA.

[1] We (arbitrarily) choose to use phrasing in terms of reward-maximization, rather than cost-minimization.

[2]Our method is a specific instantiation of a more general algorithm described therein.

[3]Available at http://alice.nc.huji.ac.il/∼tassa/

[4]We actually constrain angular velocities since limiting torque would require a stiffer integrator, but theoretical non-divergence is fully guaranteed by the viscous dissipation which enforces a Lyapunov function on the entire system, once torques are limited.

## References

[1] Remi Munos and Andrew W. Moore. Variable Resolution Discretization for High-Accuracy Solutions of Optimal Control Problems. In *International Joint Conference on Artificial Intelligence*, pages 1348–1355, 1999.

[2] M. Stilman, C. G. Atkeson, J. J. Kuffner, and G. Zeglin. Dynamic programming in reduced dimensional spaces: Dynamic planning for robust biped locomotion. In *Proceedings of the 2005 IEEE International Conference on Robotics and Automation (ICRA 2005)*, pages 2399–2404, 2005.

[3] Christopher G. Atkeson. Using local trajectory optimizers to speed up global optimization in dynamic programming. In *NIPS*, pages 663–670, 1993.

[4] C. G. Atkeson and J. Morimoto. Non-parametric representation of a policies and value functions: A trajectory based approach. In *Advances in Neural Information Processing Systems 15*, 2003.

[5] P. Abbeel, A. Coates, M. Quigley, and A. Y. Ng. An application of reinforcement learning to aerobatic helicopter flight. In *Advances in Neural Information Processing Systems 19*, 2007.

[6] J. Morimoto and C. G. Atkeson. Minimax differential dynamic programming: An application to robust bipedwalking. In *Advances in Neural Information Processing Systems 14*, 2002.

[7] Emanuel Todorov and Wei-Wei Li. Optimal control methods suitable for biomechanical systems. In *25th Annual Int. Conf. IEE Engineering in Medicine and Biology Society*, 2003.

[8] R. Munos. Policy gradient in continuous time. *Journal of Machine Learning Research*, 7:771–791, 2006.

[9] J. Peters and S. Schaal. Reinforcement learning for parameterized motor primitives. In *Proceedings of the IEEE International Joint Conference on Neural Networks (IJCNN 2006)*, 2006.

[10] Tom Erez and William D. Smart. Bipedal walking on rough terrain using manifold control. In *IEEE/RSJ International Conference on Robots and Systems (IROS)*, 2007.

[11] A. Crespi and A. Ijspeert. AmphiBot II: An amphibious snake robot that crawls and swims using a central pattern generator. In *Proceedings of the 9th International Conference on Climbing and Walking Robots (CLAWAR 2006)*, pages 19–27, 2006.

[12] D. Q. Mayne. A second order gradient method for determining optimal trajectories for non-linear discrete-time systems. *International Journal of Control*, 3:85–95, 1966.

[13] D. H. Jacobson and D. Q. Mayne. *Differential Dynamic Programming*. Elsevier, 1970.

[14] L.-Z. Liao and C. A. Shoemaker. Convergence in unconstrained discrete-time differential dynamic programming. *IEEE Transactions on Automatic Control*, 36(6):692–706, 1991.

[15] S. Yakowitz. Algorithms and computational techniques in differential dynamic programming. *Control and Dynamic Systems: Advances in Theory and Applications*, 31:75–91, 1989.

[16] L.-Z. Liao and C. A. Shoemaker. Advantages of differential dynamic programming over newton's method for discrete-time optimal control problems. Technical Report 92-097, Cornell Theory Center, 1992.

[17] E. Todorov. Iterative local dynamic programming. Manuscript under review, available at www.cogsci.ucsd.edu/~todorov/papers/ildp.pdf, 2007.

[18] S. J. Julier and J. K. Uhlmann. A new extension of the kalman filter to nonlinear systems. In *Proceedings of AeroSense: The 11th Int. Symp. on Aerospace/Defence Sensing, Simulation and Controls*, 1997.

[19] C. E. Garcia, D. M. Prett, and M. Morari. Model predictive control: theory and practice. *Automatica*, 25: 335–348, 1989.

[20] M. Stolle and C. G. Atkeson. Policies based on trajectory libraries. In *Proceedings of the International Conference on Robotics and Automation (ICRA 2006)*, 2006.

[21] R. Coulom. *Reinforcement Learning Using Neural Networks, with Applications to Motor Control*. PhD thesis, Institut National Polytechnique de Grenoble, 2002.

